# A Segment-based Automatic Language Identification System

Yeshwant K. Muthusamy & Ronald A. Cole
Center for Spoken Language Understanding
Oregon Graduate Institute of Science and Technology
Beaverton OR 97006-1999

## Abstract

We have developed a four-language automatic language identification system for high-quality speech. The system uses a neural network-based segmentation algorithm to segment speech into seven broad phonetic categories. Phonetic and prosodic features computed on these categories are then input to a second network that performs the language classification. The system was trained and tested on separate sets of speakers of American English, Japanese, Mandarin Chinese and Tamil. It currently performs with an accuracy of 89.5% on the utterances of the test set.

## 1 INTRODUCTION

Automatic language identification is the rapid automatic determination of the language being spoken, by any speaker, saying anything. Despite several important applications of automatic language identification, this area has suffered from a lack of basic research and the absence of a standardized, public-domain database of languages.

It is well known that languages have characteristic sound patterns. Languages have been described subjectively as "singsong", "rhythmic", "guttural", "nasal" etc. The key to solving the problem of automatic language identification is the detection and exploitation of such differences between languages.

We assume that each language in the world has a unique acoustic structure, and that this structure can be defined in terms of phonetic and prosodic features of speech.

Phonetic, or segmental features, include the the inventory of phonetic segments and their frequency of occurrence in speech. Prosodic information consists of the relative durations and amplitudes of sonorant (vowel-like) segments, their spacing in time, and patterns of pitch change within and across these segments.

To the extent that these assumptions are valid, languages can be identified automatically by segmenting speech into broad phonetic categories, computing segment-based features that capture the relevant phonetic and prosodic structure, and training a classifier to associate the feature measurements with the spoken language.

We have developed a language identification system that uses a neural network to segment speech into a sequence of seven broad phonetic categories. Information about these categories is then used to train a second neural network to discriminate among utterances spoken by native speakers of American English, Japanese, Mandarin Chinese and Tamil. When tested on utterances produced by six new speakers from each language, the system correctly identifies the language being spoken 89.5% of the time.

## 2    SYSTEM OVERVIEW

The following steps transform an input utterance into a decision about what language was spoken.

### Data Capture

The speech is recorded using a Sennheiser HMD 224 noise-canceling microphone, low-pass filtered at 7.6 kHz and sampled at 16 kHz.

### Signal Representations

A number of waveform and spectral parameters are computed in preparation for further processing. The spectral parameters are generated from a 128-point discrete Fourier transform computed on a 10 ms Hanning window. All parameters are computed every 3 ms.

The waveform parameters consist of estimates of (i) *zc8000*: the zero-crossing rate of the waveform in a 10 ms window, (ii) *ptp700* and *ptp8000*: the peak-to-peak amplitude of the waveform in a 10 ms window in two frequency bands (0–700 Hz and 0–8000 Hz), and (iii) *pitch*: the presence or absence of pitch in each 3 ms frame. The pitch estimate is derived from a neural network pitch tracker that locates pitch periods in the filtered (0–700 Hz) waveform [2]. The spectral parameters consist of (i) DFT coefficients, (ii) *sda700* and *sda8000*: estimates of averaged spectral difference in two frequency bands, (iii) *sdf*: spectral difference in adjacent 9 ms intervals, and (iv) *cm1000*: the center-of-mass of the spectrum in the region of the first formant.

### Broad Category Segmentation

Segmentation is performed by a fully-connected, feedforward, three-layer neural network that assigns 7 broad phonetic category scores to each 3 ms time frame of the utterance. The broad phonetic categories are: VOC (vowel), FRIC (fricative),

STOP (pre-vocalic stop), PRVS (pre-vocalic sonorant), INVS (inter-vocalic sonorant), POVS (post-vocalic sonorant), and CLOS (silence or background noise). A Viterbi search, which incorporates duration and bigram probabilities, uses these frame-based output activations to find the best scoring sequence of broad phonetic category labels spanning the utterance. The segmentation algorithm is described in greater detail in [3].

**Language Classification**

Language classification is performed by a second fully-connected feedforward network that uses phonetic and prosodic features derived from the time-aligned broad category sequence. These features, described below, are designed to capture the phonetic and prosodic differences between the four languages.

## 3    FOUR-LANGUAGE HIGH-QUALITY SPEECH DATABASE

The data for this research consisted of natural continuous speech recorded in a laboratory by 20 native speakers (10 male and 10 female) of each of American English, Mandarin Chinese, Japanese and Tamil. The speakers were asked to speak a total of 20 utterances[1]: 15 conversational sentences of their choice, two questions of their choice, the days of the week, the months of the year and the numbers 0 through 10. The objective was to have a mix of unconstrained- and restricted-vocabulary speech. The segmentation algorithm was trained on just the conversational sentences, while the language classifier used all utterances from each speaker.

## 4    NEURAL NETWORK SEGMENTATION

### 4.1    SEGMENTER TRAINING

#### 4.1.1    Training and Test Sets

Five utterances from each of 16 speakers per language were used to train and test the segmenter. The training set had 50 utterances from 10 speakers (5 male and 5 female) from each of the 4 languages, for a total of 200 utterances. The development test set had 10 utterances from a different set of 2 speakers (1 male and 1 female) from each language, for a total of 40 utterances. The final test set had 20 utterances from yet another set of 4 speakers (2 male and 2 female) from each language for a total of 80 utterances. The average duration of the utterances in the training set was 4.7 secs and that of the test sets was 5.7 secs.

#### 4.1.2    Network Architecture

The segmentation network was a fully-connected, feed-forward network with 304 input units, 18 hidden units and 7 output units. The number of hidden units was determined experimentally. Figure 1 shows the network configuration and the input features.

# NEURAL NETWORK SEGMENTATION

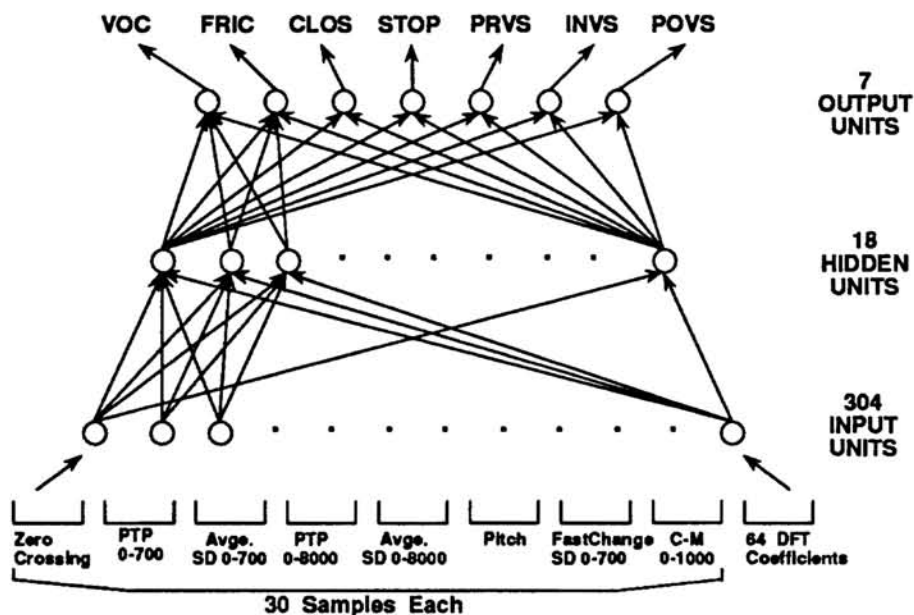

Figure 1: Segmentation Network

### 4.1.3   Feature Measurements

The feature measurements used to train the network include the 64 DFT coefficients at the frame to be classified and 30 samples each of *zc8000*, *ptp700*, *ptp8000*, *sda700*, *sda8000*, *sdf*, *pitch* and *cm1000*, for a total of 304 features. These samples were taken from a 330 ms window centered on the frame, with more samples being taken in the immediate vicinity of the frame than near the ends of the window.

### 4.1.4   Hand-labeling

Both the training and test utterances were hand-labeled with 7 broad phonetic category labels and checked by a second labeler for correctness and consistency.

### 4.1.5   Coarse Sampling of Frames

As it was not computationally feasible to train on every 3 ms frame in each utterance, only a few frames were chosen at random from each segment. To ensure approximately equal number of frames from each category, fewer frames were sampled from the more frequent categories such as vowels and closures.

### 4.1.6   Network Training

The networks were trained using backpropagation with conjugate gradient optimization [1]. Training was continued until the performance of the network on the development test set leveled off.

## 4.2  SEGMENTER EVALUATION

Segmentation performance was evaluated on the 80-utterance final test set. The segmenter output was compared to the hand-labels for each 3 ms time frame. First choice accuracy was 85.1% across the four languages. When scored on the middle 80% and middle 60% of each segment, the accuracy rose to 86.9% and 88.0% respectively, pointing to the presence of boundary errors.

# 5  LANGUAGE IDENTIFICATION

## 5.1  CLASSIFIER TRAINING

### 5.1.1  Training and Test Sets

The training set contained 12 speakers from each language, with 10 or 20 utterances per speaker, for a total of 930 utterances. The development test set contained a different group of 2 speakers per language with 20 utterances from each speaker, for a total of 160 utterances. The final test set had 6 speakers per language, with 10 or 20 utterances per speaker, for a total of 440 utterances. The average duration of the utterances in the training set was 5.1 seconds and that of the test sets was 5.5 seconds.

### 5.1.2  Feature Development

Several passes were needed through the iterative process of feature development and network training before a satisfactory feature set was obtained. Much of the effort was concentrated on statistical and linguistic analysis of the languages with the objective of determining the distinguishing characteristics among them. For example, the knowledge that Mandarin Chinese was the only monosyllabic tonal language in the set (the other three being stress languages), led us to design features that attempted to capture the large variation in pitch within and across segments for Mandarin Chinese utterances. Similarly, the presence of sequences of equal-length broad category segments in Japanese utterances led us to design an "inter-segment duration difference" feature. The final set of 80 features is described below. All the features are computed over the entire length of an utterance and use the time-aligned broad category sequence provided by the segmentation algorithm. The numbers in parentheses refer to the number of values generated.

- Intra-segment pitch variation: Average of the standard deviations of the pitch within all sonorant segments—VOC, PRVS, INVS, POVS (4 values)

- Inter-segment pitch variation: Standard deviation of the average pitch in all sonorant segments (4 values)

- Frequency of occurrence (number of occurrences per second of speech) of triples of segments. The following triples were chosen based on statistical analyses of the training data: VOC-INVS-VOC, CLOS-PRVS-VOC, VOC-POVS-CLOS, STOP-VOC-FRIC, STOP-VOC-CLOS, and FRIC-VOC-CLOS (6 values)

- Frequency of occurrence of each of the seven broad phonetic labels (7 values)

- Frequency of occurrence of all segments (number of segments per second) (1 value)
- Frequency of occurrence of all consonants (STOPs and FRICs) (1 value)
- Frequency of occurrence of all sonorants (4 values)
- Ratio of number of sonorant segments to total number of segments (1 value)
- Fraction of the total duration of the utterance devoted to each of the seven broad phonetic labels (7 values)
- Fraction of the total duration of the utterance devoted to all sonorants (1 value)
- Frequency of occurrence of voiced consonants (1 value)
- Ratio of voiced consonants to total number of consonants (1 value)
- Average duration of the seven broad phonetic labels (7 values)
- Standard deviation of the duration of the seven broad phonetic labels (7 values)
- Segment-pair ratios: conditional probability of occurrence of selected pairs of segments. The segment-pairs were selected based on histogram plots generated on the training set. Examples of selected pairs: POVS-FRIC, VOC-FRIC, INVS-VOC, etc. (27 values)
- Inter-segment duration difference: Average absolute difference in durations between successive segments (1 value)
- Standard deviation of the inter-segment duration differences (1 value)
- Average distance between the centers of successive vowels (1 value)
- Standard deviation of the distances between centers of successive vowels (1 value)

## 5.2  LANGUAGE IDENTIFICATION PERFORMANCE

### 5.2.1  Single Utterances

During the feature development phase, the 2 speakers-per-language development test set was used. The classifier performed at an accuracy of 90.0% on this small test set. For final evaluation, the development test set was combined with the original training set to form a 14 speakers-per-language training set. The performance of the classifier on the 6 speakers-per-language final test set was 79.6%. The individual language performances were English 75.8%, Japanese 77.0%, Mandarin Chinese 78.3%, and Tamil 88.0%. This result was obtained with training and test set utterances that were approximately 5.4 seconds long on the average.

### 5.2.2  Concatenated Utterances

To observe the effect of training and testing with longer durations of speech per utterance, a series of experiments were conducted in which pairs and triples of utterances from each speaker were concatenated end-to-end (with 350 ms of silence in between to simulate natural pauses) in both the training and test sets. It is to be noted that the total duration of speech used in training and testing remained unchanged for all these experiments. Table 1 summarizes the performance of the

Table 1: Percentage Accuracy on Varying Durations of Speech Per Utterance

| Avge. Duration of Training Utts. (sec) | Avge. Duration of Test Utts. (sec) | | |
|---|---|---|---|
| | 5.7 | 11.8 | 17.1 |
| 5.3 | 79.6 | 73.6 | 71.2 |
| 10.6 | 71.8 | 86.8 | 85.0 |
| 15.2 | 67.9 | 85.5 | 89.5 |

classifier when trained and tested on different durations of speech per utterance. The rows of the table show the effect of testing on progressively longer utterances for a given training set, while the columns of the table show the effect of training on progressively longer utterances for a given test set. Not surprisingly, the best performance is obtained when the classifier is trained and tested on three utterances concatenated together.

## 6   DISCUSSION

The results indicate that the system performs better on longer utterances. This is to be expected given the feature set, since the segment-based statistical features tend to be more reliable with a larger number of segments. Also, it is interesting to note that we have obtained an accuracy of 89.5% without using any spectral information in the classifier feature set. All of the features are based on the broad phonetic category segment sequences provided by the segmenter.

It should be noted that approximately 15% of the utterances in the training and test sets consisted of a fixed vocabulary: the days of the week, the months of the year and the numbers zero through ten. It is likely that the inclusion of these utterances inflated classification performance. Nevertheless, we are encouraged by the 10.5% error rate, given the small number of speakers and utterances used to train the system.

**Acknowledgements**

This research was supported in part by NSF grant No. IRI-9003110, a grant from Apple Computer, Inc., and by a grant from DARPA to the Department of Computer Science & Engineering at the Oregon Graduate Institute. We thank Mark Fanty for his many useful comments.

## Footnotes

[1]Five speakers in Japanese and one in Tamil provided only 10 utterances each.

## References

[1] E. Barnard and R. A. Cole. A neural-net training program based on conjugate-gradient optimization. Technical Report CSE 89-014, Department of Computer Science, Oregon Graduate Institute of Science and Technology, 1989.

[2] E. Barnard, R. A. Cole, M. P. Vea, and F. A. Alleva. Pitch detection with a neural-net classifier. *IEEE Transactions on Signal Processing*, 39(2):298–307, February 1991.

[3] Y. K. Muthusamy, R. A. Cole, and M. Gopalakrishnan. A segment-based approach to automatic language identification. In *Proceedings 1991 IEEE International Conference on Acoustics, Speech, and Signal Processing*, Toronto, Canada, May 1991.
